# Spectral Cues in Human Sound Localization

**Craig T. Jin**
Department of Physiology and
Department of Electrical Engineering,
Univ. of Sydney, NSW 2006, Australia

**Anna Corderoy**
Department of Physiology
Univ. of Sydney, NSW 2006, Australia

**Simon Carlile**
Department of Physiology
and Institute of Biomedical Research
Univ. of Sydney, NSW 2006, Australia

**André van Schaik**
Department of Electrical Engineering,
Univ. of Sydney, NSW 2006, Australia

## Abstract

The differential contribution of the monaural and interaural spectral cues to human sound localization was examined using a combined psychophysical and analytical approach. The cues to a sound's location were correlated on an individual basis with the human localization responses to a variety of spectrally manipulated sounds. The spectral cues derive from the acoustical filtering of an individual's auditory periphery which is characterized by the measured head-related transfer functions (HRTFs). Auditory localization performance was determined in virtual auditory space (VAS). Psychoacoustical experiments were conducted in which the amplitude spectra of the sound stimulus was varied *independently* at each ear while preserving the normal timing cues, an impossibility in the free-field environment. Virtual auditory noise stimuli were generated over earphones for a specified target direction such that there was a "false" flat spectrum at the left eardrum. Using the subject's HRTFs, the sound spectrum at the right eardrum was then adjusted so that either the true right monaural spectral cue or the true interaural spectral cue was preserved. All subjects showed systematic mislocalizations in both the true right and true interaural spectral conditions which was absent in their control localization performance. The analysis of the different cues along with the subjects' localization responses suggests there are significant differences in the use of the monaural and interaural spectral cues and that the auditory system's reliance on the spectral cues varies with the sound condition.

## 1 Introduction

Humans are remarkably accurate in their ability to localize transient, broadband noise, an ability with obvious evolutionary advantages. The study of human auditory localization has a considerable and rich history (recent review [1]) which demonstrates that there are three general classes of acoustical cues involved in the localization process: (1) interaural time differences, ITDs; (2) interaural level differences, ILDs; and (3) the spectral cues resulting

from the auditory periphery. It is generally accepted that for humans, the ITD and ILD cues only specify the location of the sound source to within a "cone of confusion" [1], i.e., a locus of points approximating the surface of a cone symmetric with respect to the interaural axis. It remains, therefore, for the localization system to extract a more precise sound source location from the spectral cues.

The utilization of the outer ear spectral cues during sound localization has been analyzed both as a statistical estimation problem, (e.g., [2]) and as optimization problem, often using neural networks, (e.g., [3]). Such computational models show that sufficient localization information is provided by the spectral cues to resolve the cone of confusion ambiguity which corroborates the psychoacoustical evidence. Furthermore, it is commonly argued that the interaural spectral cue, because of its natural robustness to level and spectral variations, has advantages over the monaural spectral cues alone. Despite these observations, there is still considerable contention as to the relative role or contribution of the monaural versus the interaural spectral cues.

In this study, each subject's spectral cues were characterized by measuring their head related transfer functions (HRTFs) for 393 evenly distributed positions in space. Measurements were carried out in an anechoic chamber and were made for both ears simultaneously using a "blocked ear" technique [1]. Sounds filtered with the HRTFs and played over earphones, which bypass the acoustical filtering of the outer ear, result in the illusion of free-field sounds which is known as virtual auditory space (VAS). The HRTFs were used to generate virtual sound sources in which the spectral cues were manipulated systematically. The recorded HRTFs along with the Glasberg and Moore cochlear model [4] were also used to generate neural excitation patterns (frequency representations of the sound stimulus within the auditory nerve) which were used to estimate the different cues available to the subject during the localization process. Using this analysis, the interaural spectral cue was characterized and the different localization cues have been correlated with each subjects' VAS localization responses.

## 2 VAS Sound Localization

The sound localization performance of four normal hearing subjects was examined in VAS using broadband white noise (300 – 14 000 Hz). The stimuli were filtered under three differing spectral conditions. (1) control: stimuli were filtered with spectrally correct left and right ear HRTFs for a given target location, (2) veridical interaural: stimuli at the left ear were made spectrally flat with an appropriate dB sound level for the given target location, while the stimuli at the right ear were spectrally shaped to preserve the correct interaural spectrum, (3) veridical right monaural: stimuli at the left ear were spectrally flat as in the second condition, while the stimuli at the right ear were filtered with the correct HRTF for the given target location, resulting in an inappropriate interaural spectral difference. For each condition, a minimum-phase filter spectral approximation was made and the interaural time difference was modeled as an all-pass delay [5]. Sounds were presented at approximately 70 dB SPL and with duration 150 ms (with 10 ms raised-cosine onset and offset ramps). Each subject performed five trials at each of 76 test positions for each stimulus condition. Detailed sound localization methods can be found in [1]. A short summary is presented below.

### 2.1 Sound Localization Task

The human localization experiments were carried out in a darkened anechoic chamber. Virtual auditory sound stimuli were presented using earphones (ER-2, Etymōtic Research, with a flat frequency response, within 3 dB, between 200–16 000 Hz). The perceived location of the virtual sound source was indicated by the subject pointing his/her nose in

the direction of the perceived source. The subject's head orientation and position were monitored using an electromagnetic sensor system (Polhemus, Inc.).

## 2.2 Human Sound Localization Performance

The sound localization performance of two subjects in the three different stimulus conditions are shown in Figure 1. The pooled data across 76 locations and five trials is presented for both the left (L) and right (R) hemispheres of space from the viewpoint of an outside observer. The target location is shown by a cross and the centroid of the subjects responses for each location is shown by a black dot with the standard deviation indicated by an ellipse. Front-back confusions are plotted, although, they were removed for calculating the standard deviations. The subjects localized the control broadband sounds accurately (Figure 1a). In contrast, the subjects demonstrated systematic mislocalizations for both the veridical interaural and veridical monaural spectral conditions (Figures 1b,c). There is clear pulling of the localization responses to particular regions of space with evident intersubject variations.

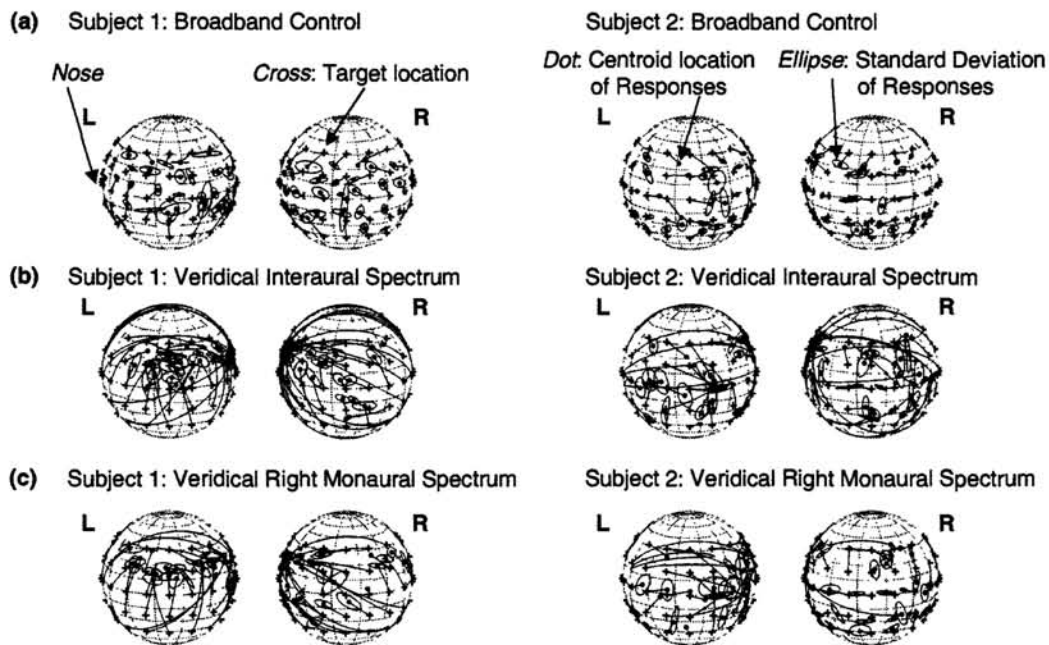

Figure 1: Localization performance for two subjects in the three sound conditions: (a) control broadband; (b) veridical interaural; (c) veridical monaural. See text for details.

## 3 Extraction of Acoustical Cues

With accurate measurements of each individual's outer ear filtering, the different acoustical cues can be compared with human localization performance on an individual basis. In order to extract the different acoustical cues in a biologically plausible manner, a model of peripheral auditory processing was used. A virtual source sound stimulus was prepared as described in Secion 2 for a particular target location. The stimulus was then filtered using a cochlear model based on the work of Glasberg and Moore [4]. This cochlear model consisted of a set of modified rounded-exponential auditory filters. The width and shape of the auditory filters change as a function of frequency (and sound level) in a manner

consistent with the known physiological and psychophysical data. These filters were logarithmically spaced on the frequency axis with a total of 200 filters between 300 Hz and 14 kHz. The cochlea's compressive non-linearity was modelled mathematically using a logarithmic function. Thus the logarithm of the output energy of a given filter indicated the amount of neural activity in that particular cochlear channel.

The relative activity across the different cochlear channels was representative of the neural excitation pattern (EP) along the auditory nerve and it is from this excitation pattern that the different spectral cues were estimated. For a given location, the left and right EPs themselves represent the monaural spectral cues. The difference in the total energy (calculated as the area under the curve) between the left and right EPs was taken as a measure of the interaural level difference and the interaural spectral shape cue was calculated as the difference between the left and right EPs. The fourth cue, interaural time difference, is a measure of the time lag between the signal in one ear as compared to the other and depends principally upon the geometrical relationship between the sound source and the human subject. This time delay was calculated using the acoustical impulse response for both ears as measured during the HRTF recordings.

## 4  Correlation of Cues and Location

For each stimulus condition and location, the acoustical cues were calculated as described above for all 393 HRTF locations. Locations at which a given cue correlates well with the stimulus cue for a particular target location were taken as analytical predictions of the subject's response locations according to that cue. As the spectral content of the signal is varied, the cue(s) available may strongly match the cue(s) normally arising from locations other than the target location. Therefore the aim of this analysis is to establish for which locations and stimulus conditions a given response most correlated with a particular cue.

The following analyses (using a Matlab toolbox developed by the authors) hinge upon the calculation of "cue correlation values". To a large extent, these calculations follow the examples described by [6] and are briefly described here. For each stimulus condition and target location, the subject performed five localizations trials. For each of the subject's five response locations, each possible cue was estimated (Section 3) assuming a flat-spectrum broadband Gaussian white noise as the stimulus. A mathematical quantity was then calculated which would give a measure of the similarity of the response location cues with the corresponding stimulus cues. The method of calculation depended on the cue and several alternative methods were tried. Generally, for a given cue, these different methods demonstrated the same basic pattern and the term "cue correlation value" has been given to the mathematical quantity that was used to measure cue similarity. The methods are as follows.

For the ITD cue, the negative of the absolute value of the difference between possible response location ITDs and the stimulus ITD was used as the ITD cue correlation values (the more positive a value, the higher its correlation). The ILD cue correlation value was calculated in a similar fashion. The cue correlation values for the left and right monaural spectral cues (in this case, the shape of the neural excitation pattern) was calculated by taking the difference between the stimulus EP and the possible response location EPs and then summing across frequency the variation of this difference about its mean value. For the interaural spectral cue, the vector difference between the left and right EPs was calculated for both the stimulus and the possible response locations. The dot product between the stimulus and the possible response location vectors gave the ISD cue correlation values.

The cue correlation values were normalized in order to facilitate meaningful comparisons across the different acoustical cues. Following Middlebrooks [6], a "z-score normalized" cue value, for each response location corresponding to a given target location, was obtained by subtracting the mean correlation value (across all possible locations) and dividing by the

standard deviation. For these new cue values, termed the cue z-score values, a score of 1.0 or greater indicates good correlation.

## 5   Relationship between the ISD and the Cone-of-Confusion

The distribution of a given cue's z-score values around the sphere of space surrounding the subject reveals the spatial directions for that cue that correlate best with the given stimulus and target location being examined. An examination of the interaural spectral cue indicated that, unlike the other cues, the range of its cue z-score variation was relatively restricted on the ipsilateral hemisphere of space relative to the sound stimulus (values on the ipsilateral side were approximately 1.0, those on the contralateral side, -1.0). This was the first indication of the more moderate variation of the ISD cue across space as compared with the monaural spectral cues.

Closer examination of the ISD cue revealed more detailed variational properties. In order to facilitate meaningful comparisons with the other cues, the ISD cue z-score values were adjusted such that all negative values (i.e., those values at locations generally contralateral to the stimulus) were set to 0.0 and the cue z-score values recalculated. The spatial distribution of the rescaled ISD cue z-score values, as compared with the cue z-score values for the other cues, is shown in Figure 2. The cone of confusion described by the ITD and ILD is clearly evident (Fig. 2a,b) and it can be seen that the ISD cue is closely aligned with these cues (Fig. 2c). Furthermore, the ISD cue demonstrates significant asymmetry along the front-back dimensions. These novel observations demonstrate that while previous work [3, 2] indicates that the ISD cue provides *sufficient* information to determine a sound's location exactly along the cone of confusion, the variation of the cue z-score values along the cone is substantially *less* than that for the monaural spectral cues (Fig. 2d), suggesting perhaps that this acts to make the monaural spectral cue a more salient cue.

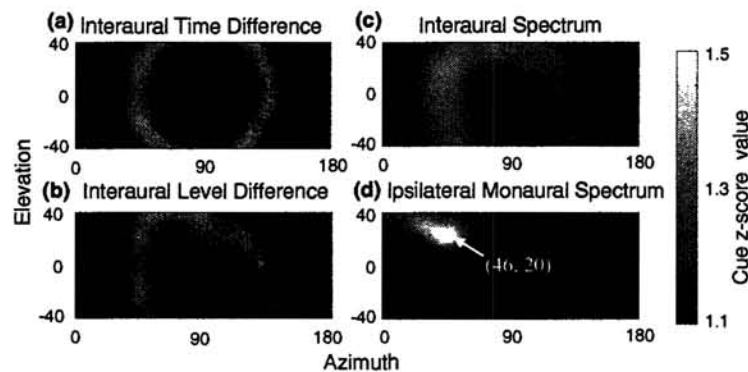

Figure 2: Spatial plot of the cue z-score values for a *single* target location (46° azimuth, 20° elevation) and broadband sound condition. Gray-scale color values indicate the cue's correlation in different spatial directions with the stimulus cue at the target location. (Z-score values for the ISD cue have been rescaled, see text.)

## 6   Analysis of Subjects Responses using Cue Z-score Values

A given cue's z-score values for the subject's responses across all 76 test locations and five trials were averaged. The mean and standard deviation are presented in a bar graph (Fig. 3). The subjects' response locations correlate highly with the ITD and ILD cue and

the standard deviation of the correlation was low (Fig. 3a,b). In other words, subjects' responses stayed on the cone of confusion of the target location. A similar analysis of the more restricted, rescaled version of the interaural spectral cue shows that despite the spectral manipulations and systematic mislocalizations, subject's were responding to locations which were highly correlated with the interaural spectral cue (Fig. 3c). The bar graphs for the monaural spectral cues ipsilateral and contralateral to the target location show the average correlation of the subjects' responses with these cues varied considerably with the stimulus condition (Fig. 3d-g) and to a lesser extent across subjects.

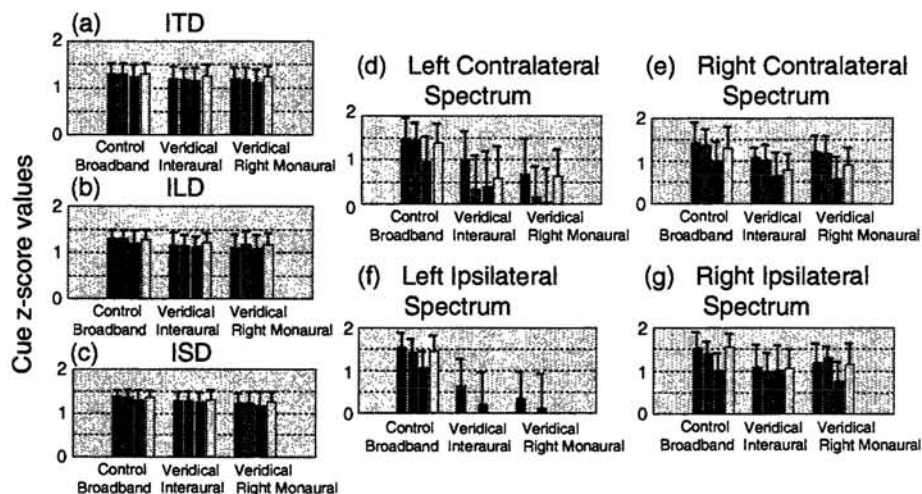

Figure 3: Correlation of the four subjects' (indicated by different gray bars) localization responses with the different acoustical cues for each stimulus condition. The bar heights indicates the mean cue z-score value, while the error bars indicate standard deviation.

# 7 Spatial Plots of Correlation Regions

As the localization responses tended to lie along the cone of confusion, the relative importance of the spectral cues along the cone of confusion was examined. The correlation values for the spectral cues associated with the subjects' responses were recalculated as a z-score value using *only* the distribution of values restricted to the cone of confusion. This demonstrates whether the spectral cues associated with the subjects' response locations were better correlated with the stimulus cues, than for any *random* location on the cone of confusion.

Spatial plots of the recalculated response cue z-score values for the spectral cues of one subject (similar trends across subjects), obtained for each stimulus location and across the three different sound conditions, is shown in Figure 4. Spatial regions of both high and low correlation are evident that vary with the stimulus spectrum. The z-score values for the ISD cue shows greater bilateral correlation across space in the veridical interaural condition (Fig. 4d) than for the veridical monaural condition (Fig. 4g), while the right monaural spectral cue demonstrates higher correlation in the right hemisphere of space for the veridical monaural condition (Fig. 4i) as opposed to the veridical interaural condition (Fig. 4f). This result (although not surprising) demonstrates that the auditory system is extracting cues to source location in a manner *dependent* on the input sound spectrum and in a manner *consistent* with the spectral information available in the sound spectrum. Figures 4e,h clearly demonstrate that the flat sound spectrum in the left ear was strongly correlated with and influenced the subject's localization judgements for specific regions of space.

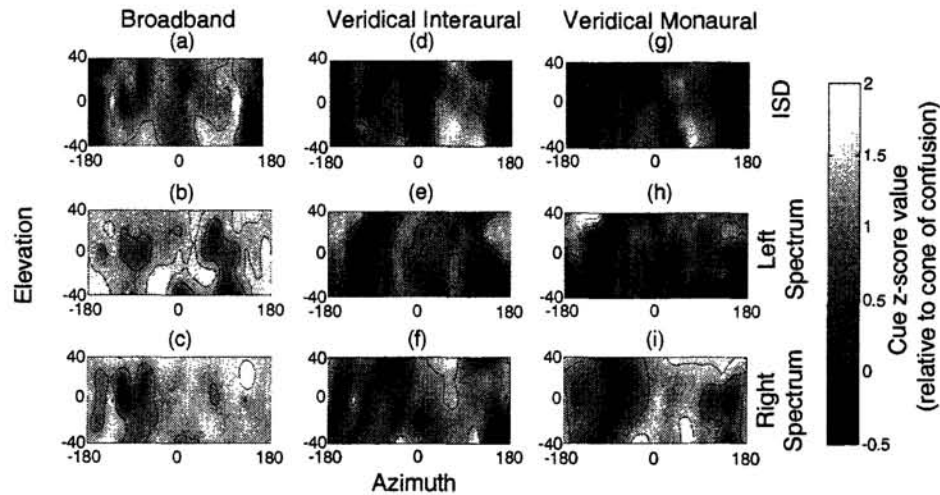

Figure 4: Spatial plot of the spectral cue z-score values for one subject's localization responses across the three different sound conditions.

## 8   Conclusions

The correlation of human sound localization responses with the available acoustical cues across three spectrally different sound conditions has provided insights into the human auditory system and its integration of cues to produce a coherent percept of spatial location. These data suggest an interrelationship between the interaural spectral cue and the cone of confusion. The ISD cue is front-back asymmetrical along the cone and its cue correlation values vary more moderately as a function of space than those of the monaural spectral cues. These data shed light on the relative role and importance of the interaural and monaural spectral cues.

### Acknowledgments

This research was supported by the ARC, NHMRC, and Dora Lush Scholarship to CJ.

## References

[1] S. Carlile, *Virtual auditory space: Generation and applications.* New York: Chapman and Hall, 1996.

[2] R. O. Duda, "Elevation dependence of the interaural transfer function," in *Binaural and spatial hearing in real and virtual environments* (R. H. Gilkey and T. R. Anderson, eds.), ch. 3, pp. 49–75, Mahwah, New Jersey: Lawrence Erlbaum Associates, 1997.

[3] J. A. Janko, T. R. Anderson, and R. H. Gilkey, "Using neural networks to evaluate the viability of monaural and interaural cues for sound localization," in *Binaural and Spatial Hearing in real and virtual environments* (R. H. Gilkey and T. R. Anderson, eds.), ch. 26, pp. 557–570, Mahwah, New Jersey: Lawrence Erlbaum Associates, 1997.

[4] B. Glasberg and B. Moore, "Derivation of auditory filter shapes from notched-noise data," *Hearing Research*, vol. 47, no. 1-2, pp. 103–138, 1990.

[5] F. Wightman and D. Kistler, "The dominant role of low-frequency interaural time differences in sound localization," *J. Acoust. Soc. Am.*, vol. 91, no. 3, pp. 1648–1661, 1992.

[6] J. Middlebrooks, "Narrow-band sound localization related to external ear acoustics," *J. Acoust. Soc. Am.*, vol. 92, no. 5, pp. 2607–2624, 1992.